# Tonal Music as a Componential Code: Learning Temporal Relationships Between and Within Pitch and Timing Components

**Catherine Stevens**
Department of Psychology
University of Queensland
QLD 4072 Australia
kates@psych.psy.uq.oz.au

**Janet Wiles**
Depts of Psychology & Computer Science
University of Queensland
QLD 4072 Australia
janetw@cs.uq.oz.au

## Abstract

This study explores the extent to which a network that *learns* the temporal relationships within and between the component features of Western tonal music can account for music theoretic and psychological phenomena such as the tonal hierarchy and rhythmic expectancies. Predicted and generated sequences were recorded as the representation of a 153-note waltz melody was learnt by a predictive, recurrent network. The network learned transitions and relations between *and* within pitch and timing components: accent and duration values interacted in the development of rhythmic and metric structures and, with training, the network developed chordal expectancies in response to the activation of individual tones. Analysis of the hidden unit representation revealed that musical sequences are represented as transitions between states in hidden unit space.

## 1 INTRODUCTION

The fundamental features of music, derivable from frequency, time and amplitude dimensions of the physical signal, can be described in terms of two systems - pitch and timing. The two systems are frequently disjoined and modeled independently of one another (e.g. Bharucha & Todd, 1989; Rosenthal, 1992). However, psychological evidence suggests that pitch and timing factors interact (Jones, 1992; Monahan, Kendall

& Carterette, 1987). The pitch and timing components can be further divided into tone, octave, duration and accent which can be regarded as a *quasi-componential* code. The important features of a *componential* code are that each component feature can be viewed as systematic in its own right (Fodor & Pylyshyn, 1988). The significance of componential codes for learning devices lies in the productivity of the system - a small (polynomial) number of training examples can generalise to an exponential test set (Brousse & Smolensky, 1989; Phillips & Wiles, 1993). We call music a *quasi-componential* code as there are significant interactions between, as well as within, the component features (we adopt this term from its use in describing other cognitive phenomena, such as reading, Plaut & McClelland, 1993).

Connectionist models have been developed to investigate various aspects of musical behaviour including composition (Hild, Feulner & Menzel, 1992), performance (Sayegh, 1989), and perception (Bharucha & Todd, 1989). The models have had success in generating novel sequences (Mozer, 1991), or developing properties characteristic of a listener, such as tonal expectancies (Todd, 1988), or reflecting properties characteristic of musical structure, such as hierarchical organisation of notes, chords and keys (Bharucha, 1992). Clearly, the models have been designed with a specific application in mind and, although some attention has been given to the representation of musical information (e.g. Mozer, 1991; Bharucha, 1992), the models rarely explain the way in which musical representations are constructed and learned. These models typically process notes which vary in pitch but are of constant duration and, as music is inherently temporal, the temporal properties of music must be reflected in both representation and processing.

There is an assumption implicit in cognitive modeling in the *information processing* framework that the representations used in any cognitive process are specified *a priori* (often assumed to be the output of a perceptual process). In the neural network framework, this view of representation has been challenged by the specification of a dual mechanism which is capable of *learning representations* and the *processes* which act upon them. Since the systematic properties of music are inherent in Western tonal music as an environment, they must also be reflected in its representation in, and processes of, a cognitive system. Neural networks provide a mechanism for learning such representations and processes, particularly with respect to temporal effects.

In this paper we study how representations can be learned in the domain of Western tonal music. Specifically, we use a recurrent network trained on a musical prediction task to construct representations of context in a musical sequence (a well-known waltz), and then test the extent to which the learned representation can account for the classic phenomena of music cognition. We see the representation construction as one aspect of learning and memory in music cognition, and anticipate that additional mechanisms of music cognition would involve memory processes that *utilise* these representations (e.g. Stevens & Latimer, 1992), although they are beyond the scope of the present paper. For example, Mozer (1992) discusses the development of higher-order, global representations at the level of relations between phrases in musical patterns. By contrast, the present study focusses on the development of representations at the level of relations between individual musical events. We expect that the additional mechanisms would, in part, develop from the behavioural aspects of music cognition that are made explicit at this early, representation construction stage.

## 2   METHOD & RESULTS

A simple recurrent network (Elman, 1989), consisting of 25 input and output units, and 20 hidden and context units, was trained according to Elman's prediction paradigm and used Backprop Through Time (BPTT) for one time step. The training data comprised the first 2 sections of *The Blue Danube* by Johann Strauss wherein each training pattern represented one event, or note, in the piece, coded as components of tone (12 values), rest (1 value), octave (3 values), duration (6 values) and accent (3 values).

In the early stages of training, the network learned to predict the prior probability of events (see Figure 1). This type of information could be encoded in the bias of the output units alone, as it is independent of temporal changes in the input patterns. After further training, the network learned to modify its predictions based on the input event (purely feedforward information) and, later still, on the context in which the event occurred (see Figure 2). Note that an important aspect of this type of recurrent network is that the representation of context is created by the network itself (Elman, 1989) and is not specified *a priori* by the network designer or the environment (Jordan, 1986). In this way, the context can encode the information in the environment which is relevant to the prediction task. Consequently, the network could, in principle, be adapted to other styles of music without modification of the design, or input and output representations.

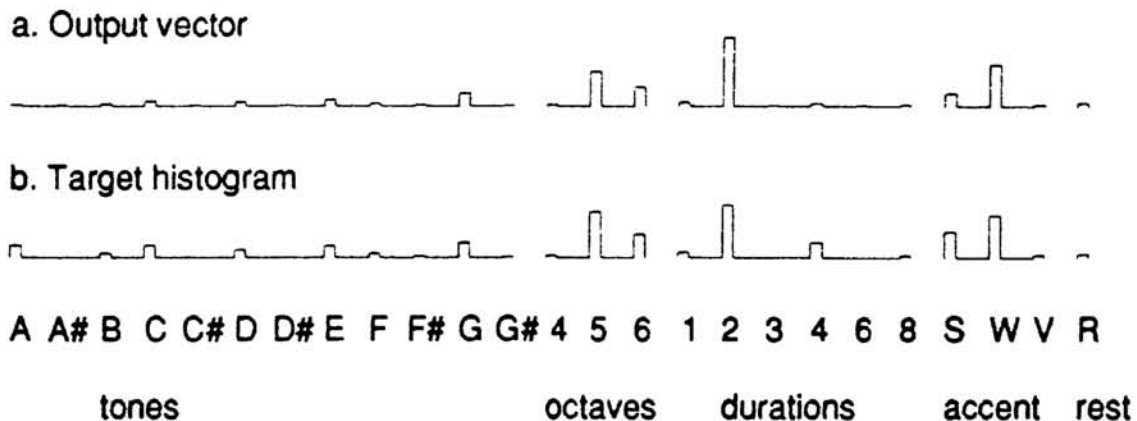

Figure 1: Comparison of output vector for the first event at Epoch 4 (a) and a histogram of the target vector averaged over all events (b). The upper graph is the predicted output of Event 1 at Epoch 4. The lower graph is a histogram of all the events in the piece, created by averaging all the target vectors. The comparison shows that the net learned initially to predict the mean target before learning the variations specific to each event.

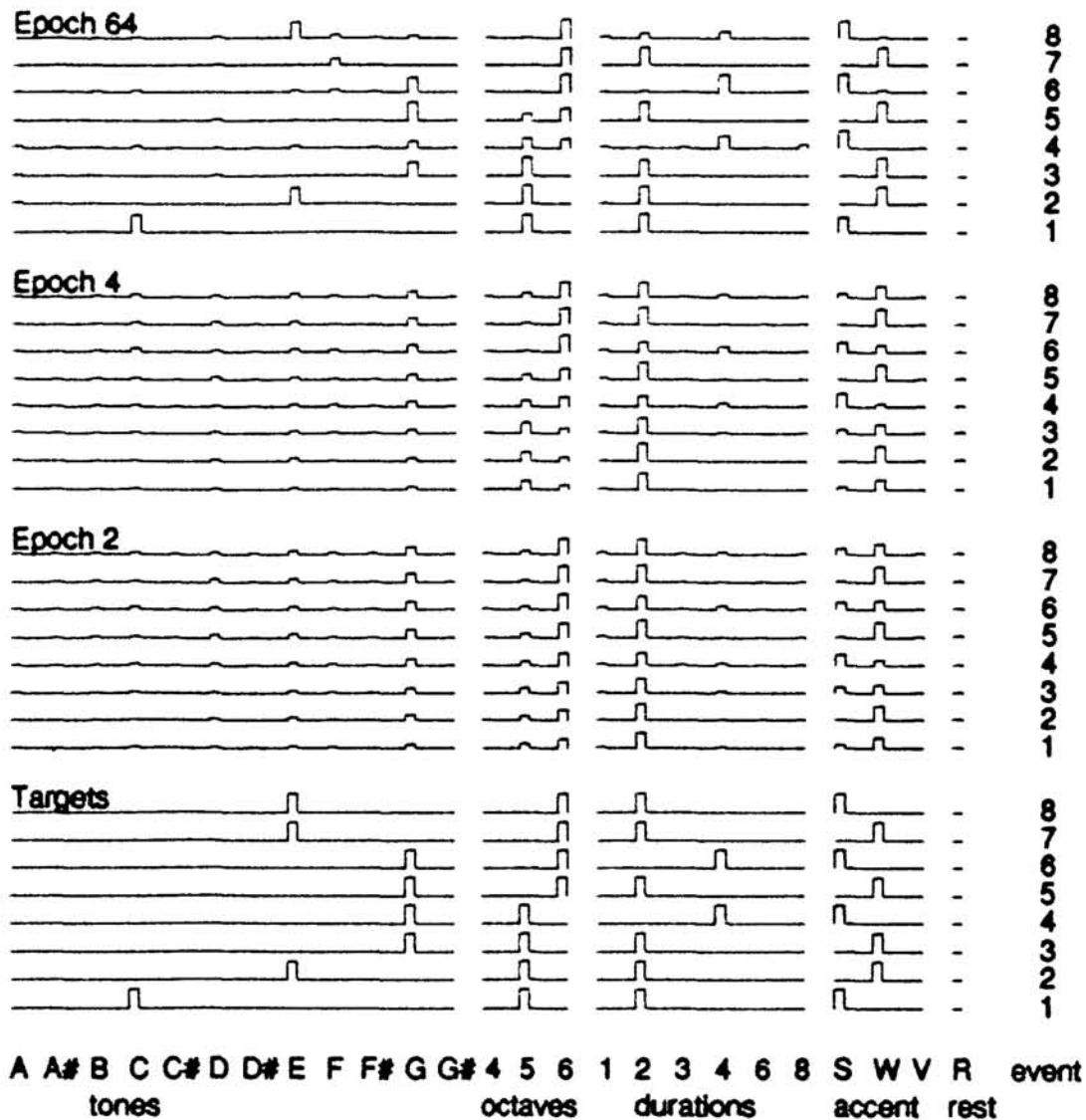

Figure 2: Evolution of the first eight events (predicted). The first block (targets) shows the correct sequence of events for the four components. In the second block (Epoch 2), the net is beginning to predict activation of strong and weak accents. In the third block (Epoch 4), the transition from one octave to another is evident. By the fourth block (Epoch 64), all four components are substantially correct. The pattern of activation across the output vector can be interpreted as a statistical description of each musical event. In psychological terms, the pattern of activation reflects the harmonic or chordal expectancies induced by each note (Bharucha, 1987) and characteristic of the tonal hierarchy (Krumhansl & Shepard, 1979).

## 3.1  NETWORK PERFORMANCE

The performance of the network as it learned to master *The Blue Danube* was recorded at log steps up to 4096 epochs. One recording comprised the output predicted by the network as the correct event was recycled as input to the network (similar to a teacher-forcing paradigm). The second recording was generated by feeding the best guess of the

output back as input.  The accent – the lilt of the waltz – was incorporated very early in the training.  Despite numerous errors in the individual events, the sequences were clearly identifiable as phrases from *The Blue Danube* and, for the most part, errors in the tone component were consistent with the tonality of the piece.  The errors are of psychological importance and the overall performance indicated that the network learnt the typical features of *The Blue Danube* and the waltz genre.

## 3.2  INTERACTIONS BETWEEN ACCENT & DURATION

Western tonal music is characterised by regularities in pitch and timing components.  For example, the occurrence of particular tones and durations in a single composition is structured and regular given that only a limited number of the possible combinations occur.  Therefore, one way to gauge performance of the network is to compare the regularities extracted and represented in the model with the statistical properties of components in the training composition.  The expected frequencies of accent-duration pairs, such as a quarter-note coupled with a strong accent, were compared with the actual frequency of occurrence in the composition: the accent and duration couplings with the highest expected frequencies were strong quarter-note (35.7), strong half-note (10.2), weak quarter-note (59.0), and weak half-note (16.9).  Scrutiny of the predicted outputs of the network over the time course of learning showed that during the initial training epochs there was a strong bias toward the event with the highest expected frequency - weak quarter-note.  The output of this accent-duration combination by the network decreased gradually.  Prediction of a strong quarter-note by the network reached a value close to the expected frequency of 35.7 by Epoch 2 (33) and then decreased gradually and approximated the actual composition frequency of 19 at Epoch 64.  Similarly, by Epoch 64, output of the most common accent and duration pairs was very close to the actual frequency of occurrence of those pairs in the composition.

## 3.3  ANALYSIS OF HIDDEN UNIT REPRESENTATIONS

An analysis of hidden unit space most often reveals structures such as regions, hierarchies and intersecting regions (Wiles & Bloesch, 1992).  In the present network, four sub-spaces would be expected (tone, octave, duration, accent), with events lying at the intersection of these sub-spaces.  A two-dimensional projection of hidden unit space produced from a canonical discriminant analysis (CDA) of duration-accent pairs reveals these divisions (see Figure 3).  In essence, there is considerable structure in the way events are represented into clusters of regions with events located at the intersection of these regions.  In Figure 3 the groups used in the CDA relate to the output values which are observable groups.  An additional CDA using groups based on position in bar showed that the hidden unit space is structured around inferred variables as well as observable ones.

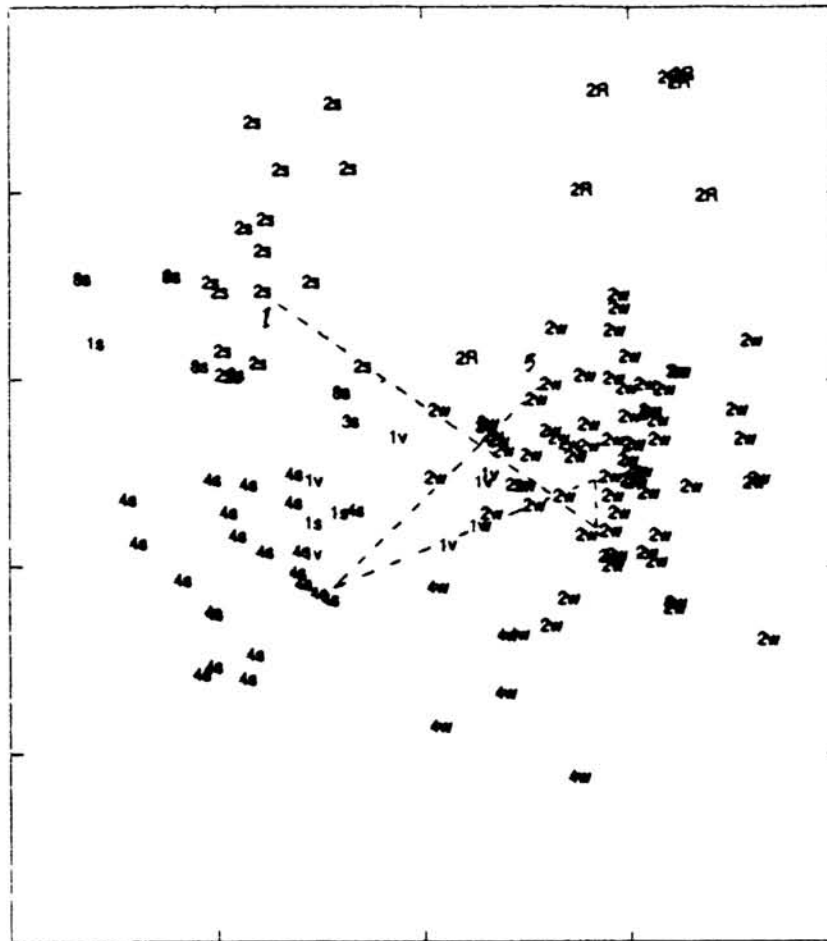

Figure 3: Two-dimensional projection of hidden unit space generated by canonical discriminant analysis of duration-accent pairs. Each note in the composition is depicted as a labelled point, and the first and third canonical components are represented along the abscissa and ordinate, respectively. The first canonical component divides strong from weak accents (denoted $s$ and $w$). In the strong ($s$) accent region of the third canonical component, quarter- and half-notes are separated (denoted by 2 and 4, respectively), and the remaining right area separates rests, weak quarter- and weak half-notes. The superimposed line shows the first five notes of the opening two bars of the composition as a trajectory through hidden unit space: there is movement along the first canonical component depending on accent ($s$ or $w$) and the second bar starts in the half-note region.

## 4    DISCUSSION & CONCLUSIONS

The focus of this study has been the extraction of information from the environment – the temporal stream of events representing *The Blue Danube* – and its incorporation into the static parameters of the weights and biases in the network. Evidence for the stages at which information from the environment is incorporated into the network representation is seen in the predicted output vectors (described above and illustrated in Figure 2). Different musical styles contain different kinds of information in the components. For example, the accent and duration components of a waltz take complementary roles in regulating the rhythm. From the durations of events alone, the position of a note in a

bar could be predicted without error. However, if the performer or listener made a single error of duration, a rhythm system based on durations alone could not recover. By contrast, accent is not a completely reliable predictor of the bar structure, but it is effective for recovery from rhythmic errors. The interaction between these two timing components provides an efficient error correction representation for the rhythmic aspect of the system. Other musical styles are likely to have similar regulatory functions performed by different components. For example, consider the use of ornaments, such as trills and mordents, in Baroque harpsichord music which, in the absence of variations in dynamics, help to signify the beat and metric structure. Alternatively, consider the interaction between pitch and timing components with the placement of harmonically-important tones at accented positions in a bar (Jones, 1992).

The network described here has learned transitions and relations between and within pitch and timing musical components and not simply the components *per se*. The interaction between accent and duration components, for example, demonstrates the manifestation of a componential code in Western tonal music. Patterns of activation across the output vector represented statistical regularities or probabilities characteristic of the composition. Notably, the representation created by the network is reminiscent of the tonal hierarchy which reflects the regularities of tonal music and has been shown to be responsible for a number of performance and memory effects observed in both musically trained and untrained listeners (Krumhansl, 1990). The distribution of activity across the tone output units can also be interpreted as chordal or harmonic expectancies akin to those observed in human behaviour by Bharucha & Stoeckig (1986). The hidden unit activations represent the rules or grammar of the musical environment; an interesting property of the simple recurrent network is that a familiar sequence can be generated by the trained network from the hidden unit activations alone. Moreover, the intersecting regions in hidden unit space represent composite states and the musical sequence is represented by transitions between states. Finally, the course of learning in the network shows an increasing specificity of predicted events to the changing context: during the early stages of training, the default output or bias of the network is towards the average pattern of activation across the entire composition but, over time, predictions are refined and become attuned to the pattern of events in particular contexts.

**Acknowledgements**
This research was supported by an Australian Research Council Postdoctoral Fellowship granted to the first author and equipment funds to both authors from the Departments of Psychology and Computer Science, University of Queensland. The authors wish to thank Michael Mozer for providing the musical database which was adapted and used in the present simulation. The modification to McClelland & Rumelhart's (1989) *bp* program was developed by Paul Bakker, Department of Computer Science, University of Queensland. The comments and suggestions made by members of the Computer Science/Psychology Connectionist Research Group at the University of Queensland are acknowledged.

**References**
Bharucha, J. J. (1987). Music cognition and perceptual facilitation: A connectionist framework. *Music Perception, 5*, 1-30.
Bharucha, J. J. (1992). Tonality and learnability. In M. R. Jones & S. Holleran (Eds.), *Cognitive bases of musical communication*, pp. 213-223. Washington: American Psychological Association.

Bharucha, J. J., & Stoeckig, K. (1986). Reaction time and musical expectancy: Priming of chords. *Journal of Experimental Psychology: Human Perception & Performance, 12*, 403-410.

Bharucha, J., & Todd, P. M. (1989). Modeling the perception of tonal structure with neural nets. *Computer Music Journal, 13*, 44-53.

Brousse, O., & Smolensky, P. (1989). Virtual memories and massive generalization in connectionist combinatorial learning. In *Proceedings of the 11th Annual Conference of the Cognitive Science Society*, pp. 380-387. Hillsdale, NJ: Lawrence Erlbaum.

Elman, J. L. (1989). *Structured representations and connectionist models.* (CRL Tech. Rep. No. 8901). San Diego: University of California, Center for Research in Language.

Fodor, J. A., & Pylyshyn, Z. W. (1988). Connectionism and cognitive architecture: A critical analysis. *Cognition, 28*, 3-71.

Hild, H., Feulner, J., & Menzel, W. (1992). HARMONET: A neural net for harmonizing chorales in the style of J. S. Bach. In J. E. Moody, S. J. Hanson & R. Lippmann (Eds.), *Advances in Neural Information Processing Systems 4*, pp. 267-274. San Mateo, CA.: Morgan Kaufmann.

Jones, M. R. (1992). Attending to musical events. In M. R. Jones & S. Holleran (Eds.), *Cognitive bases of musical communication*, pp. 91-110. Washington: American Psychological Association.

Jordan, M. I. (1986). *Serial order: A parallel distributed processing approach* (Tech. Rep. No. 8604). San Diego: University of California, Institute for Cognitive Science.

Krumhansl, C., & Shepard, R. N. (1979). Quantification of the hierarchy of tonal functions within a diatonic context. *Journal of Experimental Psychology: Human Perception & Performance, 5*, 579-594.

McClelland, J. L., & Rumelhart, D. E. (1989). *Explorations in parallel distributed processing: A handbook of models, programs and exercises.* Cambridge, Mass.: MIT Press.

Monahan, C. B., Kendall, R. A., & Carterette, E. C. (1987). The effect of melodic and temporal contour on recognition memory for pitch change. *Perception & Psychophysics, 41*, 576-600.

Mozer, M. C. (1991). Connectionist music composition based on melodic, stylistic, and psychophysical constraints. In P. M. Todd & D. G. Loy (Eds.), *Music and connectionism*, pp. 195-211. Cambridge, Mass.: MIT Press.

Mozer, M. C. (1992). Induction of multiscale temporal structure. In J. E. Moody, S. J. Hanson, & R. P. Lippmann (Eds.), *Advances in Neural Information Processing Systems 4*, pp. 275-282. San Mateo, CA.: Morgan Kaufmann.

Phillips, S., & Wiles, J. (1993). Exponential generalizations from a polynomial number of examples in a combinatorial domain. Submitted to *IJCNN*, Japan, 1993.

Plaut, D. C., & McClelland, J. L. (1993). Generalization with componential attractors: Word and nonword reading in an attractor network. To appear in *Proceedings of the 15th Annual Conference of the Cognitive Science Society.* Hillsdale, NJ: Erlbaum.

Rosenthal, D. (1992). Emulation of human rhythm perception. *Computer Music Journal, 16*, 64-76.

Sayegh, S. (1989). Fingering for string instruments with the optimum path paradigm. *Computer Music Journal, 13*, 76-84.

Stevens, C., & Latimer, C. (1991). Judgments of complexity and pleasingness in music: The effect of structure, repetition, and training. *Australian Journal of Psychology, 43*, 17-22.

Stevens, C., & Latimer, C. (1992). A comparison of connectionist models of music recognition and human performance. *Minds and Machines, 2*, 379-400.

Todd, P. M. (1988). A sequential network design for musical applications. In D. Touretzky, G. Hinton & T. Sejnowski (Eds.), *Proceedings of the 1988 Connectionist Models Summer School*, pp. 76-84. Menlo Park, CA: Morgan Kaufmann.

Wiles, J., & Bloesch, A. (1992). Operators and curried functions: Training and analysis of simple recurrent networks. In J. E. Moody, S. J. Hanson, & R. P. Lippmann (Eds.), *Advances in Neural Information Processing Systems 4.* San Mateo, CA: Morgan Kaufmann.